# Regular and Irregular Gallager-type Error-Correcting Codes

**Y. Kabashima and T. Murayama**
Dept. of Compt. Intl. & Syst. Sci.
Tokyo Institute of Technology
Yokohama 2268502, Japan

**D. Saad and R. Vicente**
Neural Computing Research Group
Aston University
Birmingham B4 7ET, UK

## Abstract

The performance of regular and irregular Gallager-type error-correcting code is investigated via methods of statistical physics. The transmitted codeword comprises products of the original message bits selected by two randomly-constructed sparse matrices; the number of non-zero row/column elements in these matrices constitutes a family of codes. We show that Shannon's channel capacity may be saturated in equilibrium for many of the regular codes while slightly lower performance is obtained for others which may be of higher practical relevance. Decoding aspects are considered by employing the TAP approach which is identical to the commonly used belief-propagation-based decoding. We show that irregular codes may saturate Shannon's capacity but with improved dynamical properties.

## 1 Introduction

The ever increasing information transmission in the modern world is based on reliably communicating messages through noisy transmission channels; these can be telephone lines, deep space, magnetic storing media etc. Error-correcting codes play a significant role in correcting errors incurred during transmission; this is carried out by encoding the message prior to transmission and decoding the corrupted received code-word for retrieving the original message.

In his ground breaking papers, Shannon[1] analyzed the capacity of communication channels, setting an upper bound to the achievable noise-correction capability of codes, given their code (or symbol) rate, constituted by the ratio between the number of bits in the original message and the transmitted code-word. Shannon's bound is non-constructive and does not provide a recipe for devising optimal codes. The quest for more efficient codes, in the hope of saturating the bound set by Shannon, has been going on ever since, providing many useful but sub-optimal codes.

One family of codes, presented originally by Gallager[2], attracted significant interest recently as it has been shown to outperform most currently used techniques[3]. Gallager-type codes are characterized by several parameters, the choice of which defines a particular member of this family of codes. Current theoretical results[3]

offer only bounds on the error probability of various architectures, proving the existence of very good codes under some restrictions; decoding issues are examined via numerical simulations.

In this paper we analyze the typical performance of Gallager-type codes for several parameter choices via methods of statistical mechanics. We then validate the analytical solution by comparing the results to those obtained by the TAP approach and via numerical methods.

## 2 The general framework

In a general scenario, a message represented by an $N$ dimensional Boolean vector $\xi$ is encoded to the $M$ dimensional vector $\boldsymbol{J}^0$ which is transmitted through a noisy channel with some flipping probability $p$ per bit (other noise types may also be studied). The received message $\boldsymbol{J}$ is then decoded to retrieve the original message.

In this paper we analyze a slightly different version of Gallager-type codes termed the MN code[3] that is based on choosing two randomly-selected sparse matrices $A$ and $B$ of dimensionality $M \times N$ and $M \times M$ respectively; these are characterized by $K$ and $L$ non-zero unit elements per row and $C$ and $L$ per column respectively. The finite numbers $K$, $C$ and $L$ define a particular code; both matrices are known to both sender and receiver. Encoding is carried out by constructing the modulo 2 inverse of $B$ and the matrix $B^{-1}A$ (mod 2); the vector $\boldsymbol{J}^0 = B^{-1}A\,\xi$ (mod 2, $\xi$ Boolean vector) constitutes the codeword. Decoding is carried out by taking the product of the matrix $B$ and the received message $\boldsymbol{J} = \boldsymbol{J}^0 + \zeta$ (mod 2), corrupted by the Boolean noise vector $\zeta$, resulting in $A\xi + B\zeta$. The equation

$$A\xi + B\zeta = AS + B\tau \quad (\text{mod } 2) \tag{1}$$

is solved via the iterative methods of Belief Propagation (BP)[3] to obtain the most probable Boolean vectors $S$ and $\tau$; BP methods in the context of error-correcting codes have recently been shown to be identical to a TAP[4] based solution of a similar physical system[5].

The similarity between error-correcting codes of this type and Ising spin systems was first pointed out by Sourlas[6], who formulated the mapping of a simpler code, somewhat similar to the one presented here, onto an Ising spin system Hamiltonian. We recently extended the work of Sourlas, that focused on extensively connected systems, to the finite connectivity case[5] as well as to the case of MN codes [7].

To facilitate the current investigation we first map the problem to that of an Ising model with finite connectivity. We employ the binary representation ($\pm 1$) of the dynamical variables $S$ and $\tau$ and of the vectors $\boldsymbol{J}$ and $\boldsymbol{J}^0$ rather than the Boolean $(0,1)$ one; the vector $\boldsymbol{J}^0$ is generated by taking products of the relevant binary message bits $J^0_\mu = \prod_{i\in\mu} \xi_i$, where the indices $\mu = \langle i_1, \ldots i_K \rangle$ correspond to the non-zero elements of $B^{-1}A$, producing a binary version of $\boldsymbol{J}^0$. As we use statistical mechanics techniques, we consider the message and codeword dimensionality ($N$ and $M$ respectively) to be infinite, keeping the ratio between them $R = N/M$, which constitutes the code rate, finite. Using the thermodynamic limit is quite natural as Gallager-type codes are usually used for transmitting long ($10^4 - 10^5$) messages, where finite size corrections are likely to be negligible. To explore the system's capabilities we examine the Hamiltonian

$$\mathcal{H} = \sum_{\mu,\sigma} \mathcal{D}_{\mu\sigma}\, \delta\left[-1 \;;\; \mathcal{J}_{\mu\sigma} \prod_{i\in\mu} S_i \prod_{j\in\sigma} \tau_j \right] - \frac{F_s}{\beta} \sum_{i=1}^{N} S_i - \frac{F_\tau}{\beta} \sum_{j=1}^{M} \tau_j \,.$$

The tensor product $\mathcal{D}_{\mu\sigma}\mathcal{J}_{\mu\sigma}$, where $\mathcal{J}_{\mu\sigma} = \prod_{i\in\mu}\xi_i\prod_{j\in\sigma}\zeta_j$ and $\sigma = \langle j_1, \ldots j_L\rangle$, is the binary equivalent of $A\boldsymbol{\xi} + B\boldsymbol{\zeta}$, treating both signal ($\boldsymbol{S}$ and index $i$) and noise ($\boldsymbol{\tau}$ and index $j$) simultaneously. Elements of the sparse connectivity tensor $\mathcal{D}_{\mu\sigma}$ take the value 1 if the corresponding indices of both signal and noise are chosen (i.e., if all corresponding indices of the matrices $A$ and $B$ are 1) and 0 otherwise; it has $C$ unit elements per $i$-index and $L$ per $j$-index representing the system's degree of connectivity. The $\delta$ function provides 1 if the selected sites' product $\prod_{i\in\mu}S_i\prod_{j\in\sigma}\tau_j$ is in disagreement with the corresponding element $\mathcal{J}_{\mu\sigma}$, recording an error, and 0 otherwise. Notice that this term is not frustrated, as there are $M+N$ degrees of freedom and only $M$ constraints from Eq.(1), and can therefore vanish at sufficiently low temperatures. The last two terms on the right represent our prior knowledge in the case of sparse or biased messages $F_s$ and of the noise level $F_\tau$ and require assigning certain values to these additive fields. The choice of $\beta \to \infty$ imposes the restriction of Eq.(1), limiting the solutions to those for which the first term of Eq.(2) vanishes, while the last two terms, scaled with $\beta$, survive. Note that the noise dynamical variables $\boldsymbol{\tau}$ are irrelevant to measuring the retrieval success $m = \frac{1}{N}\left\langle \sum_{i=1}^N \xi_i \,\text{sign}\,\langle S_i\rangle_\beta\right\rangle_\xi$ . The latter monitors the normalized mean overlap between the Bayes-optimal retrieved message, shown to correspond to the alignment of $\langle S_i\rangle_\beta$ to the nearest binary value[6], and the original message; the subscript $\beta$ denotes thermal averaging.

Since the first part of Eq.(2) is invariant under the map $S_i \to S_i\xi_i$, $\tau_j \to \tau_j\zeta_j$ and $\mathcal{J}_{\mu\sigma} \to \mathcal{J}_{\mu\sigma}\prod_{i\in\mu}\xi_i\prod_{j\in\sigma}\zeta_j = 1$, it is useful to decouple the correlation between the vectors $\boldsymbol{S}$, $\boldsymbol{\tau}$ and $\boldsymbol{\xi}$, $\boldsymbol{\zeta}$. Rewriting Eq.(2) one obtains a similar expression apart from the last terms on the right which become $F_s/\beta \sum_k S_k\,\xi_k$ and $F_\tau/\beta \sum_k \tau_k\,\zeta_k$.

The random selection of elements in $\mathcal{D}$ introduces disorder to the system which is treated via methods of statistical physics. More specifically, we calculate the partition function $\mathcal{Z}(\mathcal{D}, \boldsymbol{J}) = \text{Tr}_{\{\boldsymbol{S},\boldsymbol{\tau}\}}\exp[-\beta\mathcal{H}]$ averaged over the disorder and the statistical properties of the message and noise, using the replica method[5, 8, 9]. Taking $\beta \to \infty$ gives rise to a set of order parameters

$$q_{\alpha,\beta,\ldots,\gamma} = \left\langle \frac{1}{N}\sum_{i=1}^N Z_i\,S_i^\alpha\,S_i^\beta,\ldots,S_i^\gamma\right\rangle_{\beta\to\infty} \qquad r_{\alpha,\beta,\ldots,\gamma} = \left\langle \frac{1}{M}\sum_{i=1}^M Y_j\,\tau_j^\alpha\,\tau_j^\beta,\ldots,\tau_j^\gamma\right\rangle_{\beta\to\infty}$$

(2)

where $\alpha$, $\beta$,.. represent replica indices, and the variables $Z_i$ and $Y_j$ come from enforcing the restriction of $C$ and $L$ connections per index respectively[5]:

$$\delta\left(\sum_{\langle i_2,\ldots,i_K\rangle}\mathcal{D}_{\langle i,i_2,\ldots,j_L\rangle} - C\right) = \oint_0^{2\pi}\frac{dZ}{2\pi}\,Z^{\sum_{\langle i_2,\ldots,i_K\rangle}\mathcal{D}_{\langle i,i_2,\ldots,j_L\rangle} - (C+1)}\,,\qquad(3)$$

and similarly for the restriction on the $j$ indices.

To proceed with the calculation one has to make an assumption about the order parameters symmetry. The assumption made here, and validated later on, is that of replica symmetry in the following representation of the order parameters and the related conjugate variables

$$q_{\alpha,\beta..\gamma} = a_q\int dx\,\pi(x)\,x^l\,,\quad \widehat{q}_{\alpha,\beta..\gamma} = a_{\widehat{q}}\int d\widehat{x}\,\widehat{\pi}(\widehat{x})\,\widehat{x}^l \qquad (4)$$

$$r_{\alpha,\beta..\gamma} = a_r\int dy\,\rho(y)\,y^l\,,\quad \widehat{r}_{\alpha,\beta..\gamma} = a_{\widehat{r}}\int d\widehat{y}\,\widehat{\rho}(\widehat{y})\,\widehat{y}^l\,,$$

where $l$ is the number of replica indices, $a_*$ are normalization coefficients, and $\pi(x), \widehat{\pi}(\widehat{x}), \rho(y)$ and $\widehat{\rho}(\widehat{y})$ represent probability distributions. Unspecified integrals

are over the range $[-1, +1]$. One then obtains an expression for the free energy per spin expressed in terms of these probability distributions $1/N \langle \ln \mathcal{Z} \rangle_{\xi, \varsigma, \mathcal{D}}$ The free energy can then be calculated via the saddle point method. Solving the equations obtained by varying the free energy w.r.t the probability distributions $\pi(x), \hat{\pi}(\hat{x}), \rho(y)$ and $\hat{\rho}(\hat{y})$, is difficult as they generally comprise both delta peaks and regular[9] solutions for the ferromagnetic and paramagnetic phases (there is no spin-glass solution here as the system is not frustrated). The solutions obtained in the case of unbiased messages (the most interesting case as most messages are compressed prior to transmission) are for the ferromagnetic phase:

$$\pi(x) = \delta(x - 1) \,, \ \hat{\pi}(\hat{x}) = \delta(\hat{x} - 1) \,, \ \rho(y) = \delta(y - 1) \,, \ \hat{\rho}(\hat{y}) = \delta(\hat{y} - 1) \,, \quad (5)$$

and for the paramagnetic phase:

$$\pi(x) = \delta(x) \,, \ \hat{\pi}(\hat{x}) = \delta(\hat{x}) \,, \ \hat{\rho}(\hat{y}) = \delta(\hat{y})$$
$$\rho(y) = \frac{1 + \tanh F_\tau}{2} \delta(y - \tanh F_\tau) + \frac{1 - \tanh F_\tau}{2} \delta(y + \tanh F_\tau) \,. \quad (6)$$

These solutions obey the saddle point equations. However, it is unclear whether the contribution of other delta peaks or of an additional continuous solution will be significant and whether the solutions (5) and (6) are stable or not. In addition, it is also necessary to validate the replica symmetric ansatz itself. To address these questions we obtained solutions to the system described by the Hamiltonian (2) via TAP methods of finitely connected systems[5]; we solved the saddle point equations derived from the free energy numerically, representing all probability distributions by up to $10^4$ bin models and by carrying out the integrations via Monte-Carlo methods; finally, to show the consistency between theory and practice we carried out large scale simulations for several cases, which will be presented elsewhere.

## 3  Structure of the solutions

The various methods indicate that the solutions may be divided to two different categories: $K = L = 2$ and either $K \geq 3$ or $L \geq 3$. We therefore treat them separately.

For unbiased messages and either $K \geq 3$ or $L \geq 3$ we obtain the solutions (5) and (6) both by applying the TAP approach and by solving the saddle point equations numerically. The former was carried out at the value of $F_\tau$ which corresponds to the true noise and input bias levels (for unbiased messages $F_s = 0$) and thus to Nishimori's condition[10], where no replica symmetry breaking effects are expected. This is equivalent to having the correct prior within the Bayesian framework[6] and enables one to obtain analytic expressions for some observables as long as some gauge requirements are obeyed[10]. Numerical solutions show the emergence of stable dominant delta peaks, consistent with those of (5) and (6). The question of longitudinal mode stability (corresponding to the replica symmetric solution) was addressed by setting initial conditions for the numerical solutions close to the solutions (5) and (6), showing that they converge back to these solutions which are therefore stable.

The most interesting quantity to examine is the maximal code rate, for a given corruption process, for which messages can be perfectly retrieved. This is defined in the case of $K, L \geq 3$ by the value of $R = K/C = N/M$ for which the free energy of the ferromagnetic solution becomes smaller than that of the paramagnetic solution, constituting a first order phase transition. A schematic description of the solutions obtained is shown in the inset of Fig.1a. The paramagnetic solution ($m = 0$) has a lower free energy than the ferromagnetic one (low/high free energies are denoted

by the thick and thin lines respectively, there are no axis lines at $m = 0, 1$) for noise levels $p > p_c$ and vice versa for $p \leq p_c$; both solutions are stable. The critical code rate is derived by equating the ferromagnetic and paramagnetic free energies to obtain $R_c = 1 - H_2(p) = 1 + (p \log_2 p + (1 - p) \log_2(1 - p))$ . This coincides with *Shannon's capacity*. To validate these results we obtained TAP solutions for the unbiased message case ($K = L = 3$, $C = 6$) as shown in Fig.1a (as +) in comparison to Shannon's capacity (solid line).

Analytical solutions for the saddle point equations cannot be obtained for biased patterns and we therefore resort to numerical methods and the TAP approach. The maximal information rate (i.e., code-rate $\times H_2(f_s = (1 + \tanh F_s)/2)$ - the source redundancy) obtained by the TAP method ($\Diamond$) and numerical solutions of the saddle point equations ($\Box$), for each noise level, are shown in Fig.1a. Numerical results have been obtained using $10^3 - 10^4$ bin models for each probability distribution and had been run for $10^5$ steps per noise level point. The various results are highly consistent and practically saturate Shannon's bound for the same noise level.

The MN code for $K, L \geq 3$ seems to offer optimal performance. However, the main drawback is rooted in the co-existence of the stable $m = 1$ and $m = 0$ solutions, shown in Fig.1a (inset), which implies that from some initial conditions the system will converge to the undesired paramagnetic solution. Moreover, studying the ferromagnetic solution numerically shows a highly limited basin of attraction, which becomes smaller as $K$ and $L$ increase, while the paramagnetic solution at $m = 0$ *always* enjoys a wide basin of attraction. Computer simulations (see also [3]) show that as initial conditions for the decoding process are typically of close-to-zero magnetization (almost no prior information about the original message is assumed) it is likely that the decoding process will converge to the paramagnetic solution.

While all codes with $K, L \geq 3$ saturate Shannon's bound in their equilibrium properties and are characterized by a first order, paramagnetic to ferromagnetic, phase transition, codes with $K = L = 2$ show lower performance and different physical characteristics. The analytical solutions (5) and (6) are unstable at some flip rate levels and one resorts to solving the saddle point equations numerically and to TAP based solutions. The picture that emerges is sketched in the inset of Fig.1b: The paramagnetic solution dominates the high flip rate regime up to the point $p_1$ (denoted as 1 in the inset) in which a stable, ferromagnetic solution, of higher free energy, appears (thin lines at $m = \pm 1$). At a lower flip rate value $p_2$ the paramagnetic solution becomes unstable (dashed line) and is replaced by two stable sub-optimal ferromagnetic (broken symmetry) solutions which appear as a couple of peaks in the various probability distributions; typically, these have a lower free energy than the ferromagnetic solution until $p_3$, after which the ferromagnetic solution becomes dominant. Still, only once the sub-optimal ferromagnetic solutions disappear, at the spinodal point $p_s$, a unique ferromagnetic solution emerges as a single delta peak in the numerical results (plus a mirror solution). The point in which the sub-optimal ferromagnetic solutions disappear constitutes the maximal practical flip rate for the current code-rate and was defined numerically ($\Diamond$) and via TAP solutions (+) as shown in Fig.1b.

Notice that initial conditions for TAP and the numerical solutions were chosen almost randomly, with a slight bias of $\mathcal{O}(10^{-12})$, in the initial magnetization. The TAP dynamical equations are identical to those used for practical BP decoding[5], and therefore provide equivalent results to computer simulations with the same parameterization, supporting the analytical results. The excellent convergence results obtained point out the existence of a unique pair of global solutions to which the system converges (below $p_s$) *from practically all initial conditions*. This observation and the practical implications of using $K = L = 2$ code have not been obtained by

information theory methods (e.g.[3]); these prove the existence of very good codes for $C = L \geq 3$, and examine decoding properties only via numerical simulations.

## 4 Irregular Constructions

Irregular codes with non-uniform number of non-zero elements per column and uniform number of elements per row were recently introduced [11, 12] and were found to outperform regular codes. It is relatively straightforward to adapt our methods to study these particular constructions. The restriction of the number of connections per index can be replaced by a set of $N$ restrictions of the form (1), enforcing $C_j$ non-zero elements in the $j$-th column of the matrix $A$, and other $M$ restrictions enforcing $L_l$ non-zero elements in the $l$-th column of the matrix $B$. By construction these restrictions must obey the relations $\sum_{j=1}^{N} C_j = MK$ and $\sum_{l=1}^{M} L_l = ML$. One can assume that a particular set of restrictions is generated independently by the probability distributions $\mathcal{P}(C)$ and $\mathcal{P}(L)$. With that we can compute average properties of irregularly constructed codes generated by arbitrary distributions.

Proceeding along the same lines to those of the regular case one can find a very similar expression for the free energy which can be interpreted as a mixture of regular codes with column weights sampled with probabilities $\mathcal{P}(C)$ and $\mathcal{P}(L)$. As long as we choose probability distributions which vanish for $C, L = 0$ (avoiding trivial non-invertible matrices) and $C, L = 1$ (avoiding single checked bits), the solutions to the saddle point equations are the same as those obtained in the regular case (Eqs.5, 6) leading to exactly the same predictions for the maximum performance. The differences between regular and irregular codes show up in their dynamical behavior. In the irregular case with $K > 2$ and for biased messages the basin of attraction is larger for higher noise levels [13].

## 5 Conclusion

In this paper we examined the typical performance of Gallager-type codes. We discovered that for a certain choice of parameters, either $K \geq 3$ or $L \geq 3$, one potentially obtains optimal performance, saturating Shannon's bound. This comes at the expense of a decreasing basin of attraction making the decoding process increasingly impractical. Another code, $K = L = 2$, shows close to optimal performance with a very large basin of attraction, making it highly attractive for practical purposes. The decoding performance of both code types was examined by employing the TAP approach, an iterative method identical to the commonly used BP. Both numerical and TAP solutions agree with the theoretical results. The equilibrium properties of regular and irregular constructions is shown to be the same. The improved performance of irregular codes reported in the literature can be explained as consequence of dynamical properties. This study examines the typical performance of these increasingly important error-correcting codes, from which optimal parameter choices can be derived, complementing the bounds and empirical results provided in the information theory literature . Important aspects that are yet to be investigated include other noise types, finite size effects and the decoding dynamics itself.

**Acknowledgement** Support by the JSPS RFTF program (YK), The Royal Society and EPSRC grant GR/N00562 (DS) is acknowledged.

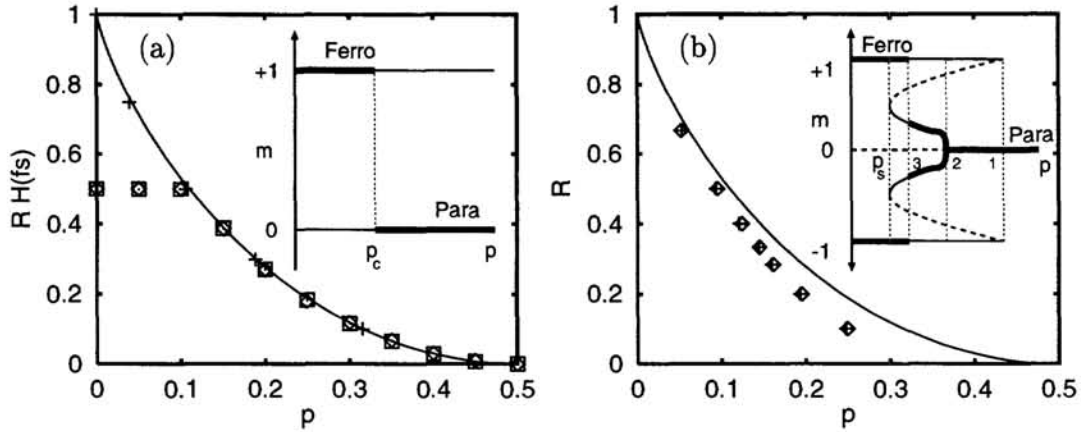

Figure 1: Critical code rate as a function of the flip rate $p$, obtained from numerical solutions and the TAP approach ($N = 10^4$), and averaged over 10 different initial conditions with error bars much smaller than the symbols size. (a) Numerical solutions for $K = L = 3$, $C = 6$ and varying input bias $f_s$ ($\square$) and TAP solutions for both unbiased ($+$) and biased ($\diamond$) messages; initial conditions were chosen close to the analytical ones. The critical rate is multiplied by the source information content to obtain the maximal information transmission rate, which clearly does not go beyond $R = 3/6$ in the case of biased messages; for unbiased patterns $H_2(f_s) = 1$. Inset: The ferromagnetic and paramagnetic solutions as functions of $p$; thick and thin lines denote stable solutions of lower and higher free energies respectively. (b) For the unbiased case of $K = L = 2$; initial conditions for the TAP ($+$) and the numerical solutions ($\diamond$) are of almost zero magnetization. Inset: The ferromagnetic (optimal/sub-optimal) and paramagnetic solutions as functions of $p$; thick and thin lines are as in (a), dashed lines correspond to unstable solutions.

# References

[1] C.E. Shannon, *Bell Sys. Tech. J.*, **27**, 379 (1948); **27**, 623 (1948).

[2] R.G. Gallager, *IRE Trans. Info. Theory*, **IT-8**, 21 (1962).

[3] D.J.C. MacKay, *IEEE Trans. IT*, **45**, 399 (1999).

[4] D. Thouless, P.W. Anderson and R.G. Palmer, *Phil. Mag.*, **35**, 593 (1977).

[5] Y. Kabashima and D. Saad, *Europhys. Lett.*, **44** 668 (1998) and **45** 97 (1999).

[6] N. Sourlas, *Nature*, **339**, 693 (1989) and *Euro. Phys. Lett.*, **25**, 159 (1994).

[7] Y. Kabashima, T. Murayama and D. Saad, *Phys. Rev. Lett.*, (1999) in press.

[8] K.Y.M. Wong and D. Sherrington, *J. Phys. A*, **20**, L793 (1987).

[9] C. De Dominicis and P. Mottishaw, *J. Phys. A*, **20**, L1267 (1987).

[10] H. Nishimori, *Prog. Theo. Phys.*, **66**, 1169 (1981).

[11] M. Luby *et. al*, *IEEE proceedings of ISIT98* (1998) and Analysis of Low Density Codes and Improved Designs Using Irregular Graphs, preprint.

[12] D.J.C. MacKay *et. al*, *IEEE Trans. Comm.*, **47**, 1449 (1999).

[13] R. Vicente *et. al*, http://xxx.lanl.gov/abs/cond-mat/9908358 (1999).
